# Maximum Likelihood Competitive Learning

**Steven J. Nowlan**[1]
Department of Computer Science
University of Toronto
Toronto, Canada
M5S 1A4

## ABSTRACT

One popular class of unsupervised algorithms are competitive algorithms. In the traditional view of competition, only one competitor, the winner, adapts for any given case. I propose to view competitive adaptation as attempting to fit a blend of simple probability generators (such as gaussians) to a set of data-points. The maximum likelihood fit of a model of this type suggests a "softer" form of competition, in which all competitors adapt in proportion to the relative probability that the input came from each competitor. I investigate one application of the soft competitive model, placement of radial basis function centers for function interpolation, and show that the soft model can give better performance with little additional computational cost.

## 1    INTRODUCTION

Interest in unsupervised learning has increased recently due to the application of more sophisticated mathematical tools (Linsker, 1988; Plumbley and Fallside, 1988; Sanger, 1989) and the success of several elegant simulations of large scale self-organization (Linsker, 1986; Kohonen, 1982). One popular class of unsupervised algorithms are competitive algorithms, which have appeared as components in a variety of systems (Von der Malsburg, 1973; Fukushima, 1975; Grossberg, 1978).

Generalizing the definition of Rumelhart and Zipser (1986), a competitive adaptive system consists of a collection of modules which are structurally identical except, possibly, for random initial parameter variation. A set of rules is defined which allow the modules to compete in some way for the right to respond to some subset

of the inputs. Typically a module is a single unit, but this need not be the case. Often, parameter restrictions are used to prevent "uninteresting" representations in which the entire set of input patterns are represented by one module.

Most of the work on competitive systems, especially within the neural network literature, has focused on a fairly extreme form of competition in which only the winner of the competition for a particular case is updated. Variants on this theme are the schemes in which, in addition to the winner, all of the losers are updated in some uniform fashion[2]. Within the statistical pattern recognition literature (Duda and Hart, 1973; McLachlan and Basford, 1988) a rather different form of competition is frequently encountered. In this form, which will be referred to as "soft" competition, all competitors are updated but the amount of update is proportional to how well each competitor did in the competition for the current case. Under a statistical model, this "soft" form of competition performs exact gradient descent in likelihood, while the more traditional winner-take-all, or "hard" competition, is an approximation to gradient descent in likelihood.

In this paper I demonstrate the superiority of "soft" competitive learning by comparing "hard" and "soft" algorithms in a classification application. The classification network consists of a layer of Radial Basis Functions (RBF's) followed by a layer of linear units which attempt to find a least mean square (LMS) fit to the desired output function (Broomhead and Lowe, 1988; Lee and Kill, 1988; Niranjan and Fallside, 1988). A network of this type can form a smooth approximation to an arbitrary function, with the RBF centers serving as control points for fitting the function (Keeler and Kowalski, 1989; Poggio and Girosi, 1989). A competitive learning component adjusts the centers of the RBF's in an unsupervised fashion, before the weights to the output units are adapted. Comparisons of hard and soft algorithms for placing the RBF's on a hand-drawn digit recognition problem and a subset of a speaker independant vowel recognition problem suggest that the soft algorithm is superior. Comparisons are also made with more traditional classifiers on the same problems.

## 2    COMPETITIVE PLACEMENT OF RBF'S

Radial Basis Function networks have been shown to be quite effective for some tasks, however a major limitation is that a very large number of RBF's may be required in high dimensional spaces. One method for using RBF's places the centers of the RBF's at the interstices of some coarse lattice defined over the input space (Broomhead and Lowe, 1988). If we assume the lattice is uniform with $k$ divisions along each dimension, and the dimensionality of the input space is $d$, a uniform lattice would require $k^d$ RBF's. This exponential growth makes the use of such a uniform lattice impractical for any high dimensional space. Another choice is to center the RBF's on the first $n$ training samples, but this method is subject to sampling error,

and a very large number of samples can be required to adequately represent the distribution of inputs. This is particularly true in high dimensional spaces where it is extremely difficult to visualize the input distribution and determine whether the training examples adequately represent this distribution.

Moody and Darken (1988) have suggested a method in which a much smaller number of RBF's are used, however the centers of these RBF's are allowed to adapt to the input samples, so they learn to represent only the part of the input space actually represented by the data. The adaptive strategy also allows the center of each RBF to be determined by a large number of training samples, greatly reducing sampling error. In their method, an unsupervised algorithm (a version of k-means) is used to select the centers of the RBF's and some *ad hoc* heuristics are suggested for adjusting the size of the RBF's to get a smooth interpolator. The weights from the hidden to the output layer are adapted to minimize a Least Mean Square (LMS) criterion. Moody and Darken were able to attain performance levels equivalent to a multi-layer Back Propagation network on a chaotic time series prediction task and a vowel discrimination task. Significant savings in training time were also reported.

The k-means algorithm used by Moody and Darken can be easily reformulated as a form of competitive adaptation. In the basic k-means algorithm (Duda and Hart, 1973) the training samples are first assigned to the class of the closest mean. The means are then recomputed as the average of the samples in their class. This two step process is repeated until the means stop changing. This is simply the "batch" version of a competitive learning scheme in which the activity of each competing unit is proportional to the distance between its weight vector and the current input vector, and the winning unit on each case adapts by adding a portion of the current input to its weight vector (with appropriate normalization).

We will now consider a statistical formalization of a competitive process for placing the centers of RBF's. Let each competing unit represent a radially symmetric (spherical) gaussian probability distribution, with the weight vector of the unit $\vec{\mu}_j$ representing the center or mean of the gaussian. The probability that the gaussian associated with unit $j$ generated an input vector $\vec{x}_k$ is

$$p(\vec{x}_k) = \frac{1}{K\sigma_j} e^{-\frac{(x_k - \mu_j)^2}{2\sigma_j^2}} \tag{1}$$

where $K$ is a normalization constant, and the covariance matrix is $\sigma_j^2 I$.

A collection of M such units is a model of the input distribution. The parameters of these M gaussians can be adjusted so that the overall average likelihood of generating the training examples is maximized. The likelihood of generating a set of observations $\{\vec{x}_1, \vec{x}_2, \ldots, \vec{x}_n\}$ from the current model is

$$L = \prod_k P(\vec{x}_k) \tag{2}$$

where $P(\vec{x}_k)$ is the probability of generating observation $\vec{x}_k$ under the current model. (For mathematical convenience we usually work with $\log L$.) If gaussian $i$ is selected

with probability $\pi_i$ and a sample is drawn from the selected gaussian, the probability of observing $\vec{x}_k$ is

$$P(\vec{x}_k) = \sum_{i=1}^{N} \pi_i \, p_i(\vec{x}_k) \qquad (3)$$

where $p_i(\vec{x}_k)$ is the probability of observing $\vec{x}_k$ under gaussian distribution $i$. The summation in (3) is awkward to work with, and frequently one of the $p_i(\vec{x}_k)$ is much larger than any of the others. Therefore, a convenient approximation for (3) is

$$P(\vec{x}_k) = \text{MAX}_{i=1}^{N} \pi_i \, p_i(\vec{x}_k) \qquad (4)$$

This is equivalent to assigning all of the responsibility for an observation to the gaussian with the highest probability of generating that observation. This approximation is frequently referred to as the "winner-take-all" assumption. It may also be regarded as a "hard" competitive decision among the gaussians. When we use (3) directly, all of the gaussians share responsibility for each observation in proportion to their probability of generating the observation. This sharing of responsibility can be regarded as a "soft" competitive decision among the gaussians.

The maximum likelihood estimate for the mean of each gaussian in our model can be found by evaluating $\partial \log L / \partial \vec{\mu}_j = 0$. We will consider a simple model in which we assume that $\pi_j$ and $\sigma_j$ are the same for all of the gaussians, and compare the hard and soft estimates for $\vec{\mu}_j$.

With the hard approximation, substituting (4) in (2), the maximum likelihood estimate of $\vec{\mu}_j$ has the simple form

$$\hat{\vec{\mu}}_j = \frac{\sum_{k \in C_j} \vec{x}_k}{N_j} \qquad (5)$$

where $C_j$ is the set of cases closest to gaussian $j$, and $N_j$ is the size of this set. This is identical to the expression for $\vec{\mu}_j$ in the k-means algorithm.

Rather than using the approximation in (4) we can find the exact maximum likelihood estimates for $\vec{\mu}_j$ by substituting (3) in (2). The estimate for the mean is now

$$\hat{\vec{\mu}}_j = \frac{\sum_k p(j|\vec{x}_k)\vec{x}_k}{\sum_k p(j|\vec{x}_k)} \qquad (6)$$

where $p(j|\vec{x}_k)$ is the probability, given that we have observed $\vec{x}_k$, of gaussian $j$ having generated $\vec{x}_k$. For the simple model used here

$$p(j|\vec{x}_k) = \frac{p_j(\vec{x}_k)}{\sum_{i=1}^{M} p_i(\vec{x}_k)}$$

Comparing (6) and (5), the hard competitive model uses the average of the cases unit $j$ is closest to in recomputing its mean, while the soft competitive model uses the average of all the cases weighted by $p(j|\vec{x}_k)$.

We can use either the approximate or exact likelihood algorithm to position the RBF's in an interpolation network. If $\vec{x}_k$ is the current input, each RBF unit computes $p_j(\vec{x}_k)$ as its output activation $a_j$. For the hard competitive model, a winner-take-all operation then sets $a_j = 1$ for the most active unit and $a_i = 0$ for all other units. Only the winning unit will update its mean vector, and for this update we use the iterative version of (5). In the soft competitive model we normalize each $a_j$ by dividing it by the sum of $a_j$ over all RBF's. In this case the mean vectors of all of the hidden units are updated according to the iterative version of (6). The computational cost difference between the winner-take-all operation in the hard model and the normalization in the soft model is negligible; however, if the algorithms are implemented sequentially, the soft model requires more computation because all of the means, rather than just the mean of the winner, are updated for each case.

The two models described in this section are easily extended to allow each spherical gaussian to have a different variance $\sigma_j^2$. The activation of each RBF unit is now a function of $(\vec{x}_k - \vec{\mu}_j)/\sigma_j$, but the expressions for the maximum likelihood estimates of $\vec{\mu}_j$ are the same. Expressions for updating $\sigma_j^2$ can be found by solving $\partial \log L/\partial \sigma_j^2 = 0$. Some simulations have also been performed with a network in which each RBF had a diagonal covariance matrix, and each of the $d$ variance components was estimated separately (Nowlan, 1990).

## 3   APPLICATION TO TWO CLASSIFICATION TASKS

The architecture described above was used for a digit classification and a vowel discrimination task. The networks were trained by first using the soft or hard competitive algorithm to determine the means and variances of the RBF's, and, once these were learned, then training the output layer of weights. The weights from the RBF's to the output layer were trained using a recursive least squares algorithm, allowing an exact LMS solution to be found with one pass through the training set. (A target of $+1$ was used for the correct output category and $-1$ for all of the other categories.) For the hard competitive model the unnormalized probabilities $p_j(\vec{x})$ were used as the RBF unit outputs, while the soft competitive model used the normalized probabilities $p(j|\vec{x})$.

The first task required the classification of a set of hand drawn digits from 12 subjects. There were 480 input patterns, divided into 320 training patterns and 160 testing patterns, with examples from all subjects in both groups. Each pattern was digitized on a 16 by 16 grid. These 256 dimensional binary vectors were used as input to the classification network, and there were 10 output units.

Networks with 40 and 150 spherical gaussians were simulated. Both hard and soft algorithms were used with all configurations. The performance of these networks on the testing set is summarized in Table 1. This table also contains performance results for a multi-layer back propagation network, a two layer linear network, and a nearest neighbour classifier on the same task. The nearest neighbour classifier used all 320 labeled training samples and based its decision on the class of the

| Type of Classifier | % Correct on Test Set |
|---|---|
| 40 Sph. Gauss. – Hard | 87.6% |
| 40 Sph. Gauss. – Soft | 91.8% |
| 150 Sph. Gauss. – Hard | 90.1% |
| 150 Sph. Gauss. – Soft | 94.0% |
| Layered BP Net | 94.5% |
| Linear Net | 60.0% |
| Nearest Neighbour | 83.1% |

**Table 1:** Summary of Performance for Digit Classification

nearest neighbour only[3]. The relatively poor performance of the nearest neighbour classifier is one indication of the difficulty of this task. The two layer linear network was trained with a recursive least squares algorithm[4]. The back propagation network was developed specifically for this task (le Cun, 1987), and used a specialized architecture with three layers of hidden units, localized receptive fields, and weight sharing to reduce the number of free parameters in the system.

Table 1 reveals that the networks were trained using the soft competitive algorithm to determine means and variances of the RBF's were superior in performance to identical networks trained with the hard competitive algorithm. The RBF network using 150 spherical gaussians was able to equal the performance level of the sophisticated back propagation network, and a network with 40 spherical RBF's performed considerably better than the nearest neighbour classifier.

The second task was a speaker independent vowel recognition task. The data consisted of a digitized version of the first and second formant frequencies of 10 vowels for multiple male and female speakers (Peterson and Barney, 1952). Moody and Darken (1988) have previously applied to this data an architecture which is very similar to the one suggested here, and Huang and Lippmann (1988) have compared the performance of a number of different classifiers on this same data. More recently, Bridle (1989) has applied a supervised algorithm which uses a "softmax" output function to this data. This softmax function is very similar to the equation for $p(j|\vec{x}_k)$ used in the soft competitive model. The results from these studies are included in Table 2 along with the results for RBF networks using both hard and soft competition to determine the RBF parameters. All of the classifiers were trained on a set of 338 examples and tested on a separate set of 333 examples.

As with the digit classification task, the RBF networks trained using the soft adaptive procedure show uniformly better performance than equivalent networks trained using the hard adaptive procedure. The results obtained for the hard adaptive pro-

| Type of Classifier | % Correct on Test Set |
|---|---|
| 20 Sph. Gauss. – Hard | 75.1% |
| 20 Sph. Gauss. – Soft | 82.6% |
| 100 Sph. Gauss. – Hard | 82.6% |
| 100 Sph. Gauss. – Soft | 87.1% |
| 20 RBF's (Moody *et al*) | 73.3% |
| 100 RBF's (Moody *et al*) | 82.0% |
| K Nearest Neighbours (Lippmann *et al*) | 82.0% |
| Gaussian Classifier (Lippmann *et al*) | 79.7% |
| 2 Layer BP Net (Lippmann *et al*) | 80.2% |
| Feature Map (Lippmann *et al*) | 77.2% |
| 2 Layer Softmax (Bridle) | 78.0% |

Table 2: Summary of Performance for Vowel Classification

cedure with 20 and 100 spherical gaussians are very close to Moody and Darken's results, which is expected since the procedures are identical except for the manner in which the variances are obtained. Table 2 also reveals that the RBF network with 100 spherical gaussians, trained with the soft adaptive procedure, performed better than any of the other classifiers that have been applied to this data.

## 4   DISCUSSION

The simulations reported in the previous section provide strong evidence that the exact maximum likelihood (or soft) approach to determining the centers and sizes of RBF's leads to better classification performance than the winner-take-all approximation. In both tasks, for a variety of numbers of RBF's, the exact maximum likelihood approach outperformed the approximate method. Comparing (5) and (6) reveals that this improved performance can be obtained with little additional computational burden.

The performance of the RBF networks on these two classification tasks also shows that hybrid approaches which combine unsupervised and supervised procedures are capable of competent levels of performance on difficult problems. In the digit classification task the hybrid RBF network was able to equal the performance level of a sophisticated multi-layer supervised network, while in the vowel recognition task the hybrid network obtained the best performance level of any of the classification networks. One reason why the hybrid model is interesting is that since the hidden unit representation is independent of the classification task, it may be used for many different tasks without interference between the tasks. (This is actually demonstrated in the simulations described, since each category in the two tasks can be regarded as a separate classification problem.) Even if we are only interested in using the network for one task, there are still advantages to the hybrid approach. In many domains, such as speech, unlabeled samples can be obtained much more

cheaply than labeled samples. To avoid over-fitting, the amount of training data must generally be considerably greater than the number of free parameters in the model. In the hybrid models, especially in high dimensional input spaces, most of the parameters are in the unsupervised part of the model[5]. The unsupervised stage may be trained with a large body of unlabeled samples, and a much smaller body of labeled samples can be used to train the output layer.

The performance on the digit classification task also shows that RBF networks can deal effectively with tasks with high (256) dimensional input spaces and highly non-gaussian input distributions. The competitive network was able to succeed on this task with a relatively small number of RBF's because the data was actually distributed over a much lower dimensional subspace of the input space. The soft competitive network automatically concentrates its representation on this subspace, and in this fashion performs a type of implicit dimensionality reduction. Moody (1989) has also mentioned this type of dimensionality reduction as a factor in the success of some of the models he has worked with.

The success of the soft adaptive strategy in these interpolation networks encourages one to extend the soft interpretation in other directions. The feature maps of Kohonen (1982) incorporate a hard competitive process, and a soft version of the feature map algorithm could be developed. In addition, there is a class of decision-directed, or "bootstrap", learning algorithms which use their own outputs to provide a training signal. These algorithms can be regarded as hard competitive processes, and new algorithms which use the soft assumption may be developed from the bootstrap procedure (Nowlan and Hinton, 1989). Bridle (1989) has suggested a different type of output unit for supervised networks, which incorporates the idea of a "softmax" type of competition. Finally, the maximum likelihood approach is easily extended to non-gaussian models, and one model of particular interest would be the Boltzmann machine.

## Acknowledgements

I would like to thank Richard Lippmann of Lincoln Laboratories and John Moody of Yale University for making the vowel formant data available to me. I would also like to thank Geoff Hinton, and the members of the Connectionist Research Group of the University of Toronto, for many helpful comments and suggestions while conducting this research and preparing this paper.

## Footnotes

[1] The author is visiting the University of Toronto while completing a PhD at Carnegie Mellon University.

[2]The feature maps of Kohonen (1982) are actually a special case in which a few units are adapted at once, however the units which are adapted in addition to the winner are selected by a neighbourhood function rather than by how well they represent the current data.

[3] Two, three, and five nearest neighbour classifiers were also tried, but they all performed worse than nearest neighbour.

[4] This network was included to show that the linear layer is not doing all of the work in the hybrid RBF networks.

[5]In the digit task, there are over 25 times as many parameters in the unsupervised part of the network as there are in the supervised part.

## References

Bridle, J. (1989). Probabilistic interpretation of feedforward classification network outputs, with relationships to statistical pattern recognition. In Fougelman-Soulie, F. and Herault, J., editors, *Neuro-computing: algorithms, architectures and applications.* Springer-Verlag.

Broomhead, D. and Lowe, D. (1988). Multivariable functional interpolation and adaptive networks. *Complex Systems*, 2:321–355.

Duda, R. and Hart, P. (1973). *Pattern Classification And Scene Analysis.* Wiley and Son.

Fukushima, K. (1975). Cognitron: A self-organizing multilayered neural network. *Biological Cybernetics*, 20:121–136.

Grossberg, S. (1978). A theory of visual coding, memory, and development. In *Formal theories of visual perception*. John Wiley and Sons, New York.

Huang, W. and Lippmann, R. (1988). Neural net and traditional classifiers. In Anderson, D., editor, *Neural Information Processing Systems*. American Institute of Physics.

Keeler, E. H. J. and Kowalski, J. (1989). Layered neural networks with gaussian hidden units as universal approximators. MCC Technical Report ACT-ST-272-89, MCC.

Kohonen, T. (1982). Self-organized formation of topologically correct feature maps. *Biological Cybernetics*, 43:59–69.

le Cun, Y. (1987). *Modèles Connexionnistes de l'Apprentissage*. PhD thesis, Université Pierre et Marie Curie, Paris, France.

Lee, S. and Kill, R. (1988). Multilayer feedforward potential function networks. In *Proceedings IEEE Second International Conference on Neural Networks*, page I:161, San Diego, California.

Linsker, R. (1986). From basic network principles to neural architecture: Emergence of spatial opponent cells. *Proceedings of the National Academy of Sciences USA*, 83:7508–7512.

Linsker, R. (1988). Self-organization in a perceptual network. *IEEE Computer Society*, pages 105–117.

McLachlan, G. and Basford, K. (1988). *Mixture Models: Inference and Applications to Clustering*. Marcel Dekker, New York.

Moody, J. (1989). Fast learning in multi-resolution hierarchies. Technical Report YALEU/DCS/RR-681, Yale University.

Moody, J. and Darken, C. (1988). Learning with localized receptive fields. In D. Touretzky, G. Hinton, T. S., editor, *Proceedings of the 1988 Connectionist Models Summer School*, pages 133–143. Morgan Kauffman.

Niranjan, M. and Fallside, F. (1988). Neural networks and radial basis functions in classifying static speech patterns. Technical Report CUEDIF-INFENG17R22, Engineering Dept., Cambridge University. to appear in Computers Speech and Language.

Nowlan, S. (1990). Maximum likelihood competition in RBF networks. Technical Report CRG-TR-90-2, University of Toronto Connectionist Research Group.

Nowlan, S. and Hinton, G. (1989). Maximum likelihood decision-directed adaptive equalization. Technical Report CRG-TR-89-8, University of Toronto Connectionist Research Group.

Peterson, G. and Barney, H. (1952). Control methods used in a study of vowels. *The Journal of the Acoustical Society of America*, 24:175–184.

Plumbley, M. and Fallside, F. (1988). An information theoretic approach to unsupervised connectionist models. In D. Touretzky, G Hinton, T. S., editor, *Proceedings of the 1988 Connectionist Models Summer School*, pages 239–245. Morgan Kauffmann.

Poggio, G. and Girosi, F. (1989). A theory of networks for approximation and learning. A.I. Memo 1140, MIT.

Rumelhart, D. E. and Zipser, D. (1986). Feature discovery by competitive learning. In *Parallel distributed processing: Explorations in the microstructure of cognition*, volume I. Bradford Books, Cambridge, MA.

Sanger, T. (1989). An optimality principle for unsupervised learning. In Touretzky, D., editor, *Advances in Neural Information Processing Systems 1*, pages 11–19. Morgan Kauffman.

Von der Malsburg, C. (1973). Self-organization of orientation sensitive cells in striate cortex. *Kybernetik*, 14:85–100.
